# Using Curvature Information for Fast Stochastic Search

**Genevieve B. Orr**
Dept of Computer Science
Willamette University
900 State Street
Salem, OR 97301
gorr@willamette.edu

**Todd K. Leen**
Dept of Computer Science and Engineering
Oregon Graduate Institute of
Science and Technology
P.O.Box 91000, Portland, Oregon 97291-1000
tleen@cse.ogi.edu

## Abstract

We present an algorithm for fast stochastic gradient descent that uses a nonlinear adaptive momentum scheme to optimize the late time convergence rate. The algorithm makes effective use of curvature information, requires only $\mathcal{O}(n)$ storage and computation, and delivers convergence rates close to the theoretical optimum. We demonstrate the technique on linear and large nonlinear back-prop networks.

## Improving Stochastic Search

Learning algorithms that perform gradient descent on a cost function can be formulated in either stochastic (on-line) or batch form. The stochastic version takes the form

$$\omega_{t+1} = \omega_t + \mu_t\, G(\omega_t, x_t) \qquad (1)$$

where $\omega_t$ is the current weight estimate, $\mu_t$ is the learning rate, $G$ is minus the instantaneous gradient estimate, and $x_t$ is the input at time $t$[1]. One obtains the corresponding batch mode learning rule by taking $\mu$ constant and averaging $G$ over all $x$.

Stochastic learning provides several advantages over batch learning. For large datasets the batch average is expensive to compute. Stochastic learning eliminates the averaging. The stochastic update can be regarded as a noisy estimate of the batch update, and this intrinsic noise can reduce the likelihood of becoming trapped in poor local optima [1, 2].

The noise must be reduced late in the training to allow weights to converge. After settling within the basin of a local optimum $\omega_*$, learning rate annealing allows convergence of the weight error $v \equiv \omega - \omega_*$. It is well-known that the expected squared weight error, $E[\|v\|^2]$ decays at its maximal rate $\propto 1/t$ with the annealing schedule $\mu_0/t$. Furthermore to achieve this rate one must have $\mu_0 > \mu_{crit} = 1/(2\lambda_{min})$ where $\lambda_{min}$ is the smallest eigenvalue of the Hessian at $\omega_*$ [3, 4, 5, and references therein]. Finally the *optimal* $\mu_0$, which gives the lowest possible value of $E[\|v\|^2]$ is $\mu_0 = 1/\lambda$. In multiple dimensions the optimal *learning rate matrix* is $\mu(t) = (1/t)\mathcal{H}^{-1}$, where $\mathcal{H}$ is the Hessian at the local optimum.

Incorporating this curvature information into stochastic learning is difficult for two reasons. First, the Hessian is not available since the point of stochastic learning is *not* to perform averages over the training data. Second, even if the Hessian were available, optimal learning requires its *inverse* – which is prohibitively expensive to compute [2].

The primary result of this paper is that one can achieve an algorithm that behaves optimally, i.e. as if one had incorporated the inverse of the full Hessian, without the storage or computational burden. The algorithm, which requires only $\mathcal{O}(n)$ storage and computation ($n$ = number of weights in the network), uses an adaptive momentum parameter, extending our earlier work [7] to fully non-linear problems. We demonstrate the performance on several large back-prop networks trained with large datasets.

Implementations of stochastic learning typically use a constant learning rate during the early part of training (what Darken and Moody [4] call the search phase) to obtain exponential convergence towards a local optimum, and then switch to annealed learning (called the converge phase). We use Darken and Moody's adaptive search then converge (ASTC) algorithm to determine the point at which to switch to $1/t$ annealing. ASTC was originally conceived as a means to insure $\mu_0 > \mu_{crit}$ during the annealed phase, and we compare its performance with adaptive momentum as well. We also provide a comparison with conjugate gradient optimization.

## 1 Momentum in Stochastic Gradient Descent

The adaptive momentum algorithm we propose was suggested by earlier work on convergence rates for annealed learning with constant momentum. In this section we summarize the relevant results of that work.

Extending (1) to include momentum leaves the learning rule

$$\omega_{t+1} = \omega_t + \mu_t\, G(\omega_t, x_t) + \beta\, (\omega_t - \omega_{t-1}) \tag{2}$$

where $\beta$ is the momentum parameter constrained so that $0 < \beta < 1$. Analysis of the dynamics of the expected squared weight error $E[\|v\|^2]$ with $\mu_t = \mu_0/t$ learning rate annealing [7, 8] shows that at late times, learning proceeds as for the algorithm without momentum, but with a scaled or *effective* learning rate

$$\mu_{eff} \equiv \frac{\mu_0}{1 - \beta}\,. \tag{3}$$

This result is consistent with earlier work on momentum learning with small, *constant* $\mu$, where the same result holds [9, 10, 11]

If we allow the effective learning rate to be a matrix, then, following our comments in the introduction, the *lowest value* of the misadjustment is achieved when $\mu_{eff} = \mathcal{H}^{-1}$ [7, 8]. Combining this result with (3) suggests that we adopt the heuristic[3]

$$\beta_{opt} = I - \mu_0 \mathcal{H}. \tag{4}$$

where $\beta_{opt}$ is a *matrix* of momentum parameters, $I$ is the identity matrix, and $\mu_0$ is a scalar.

We started with a scalar momentum parameter constrained by $0 < \beta < 1$. The equivalent constraint for our matrix $\beta_{opt}$ is that its eigenvalues lie between 0 and 1. Thus we require $\mu_0 < 1/\lambda_{max}$ where $\lambda_{max}$ is the largest eigenvalue of $\mathcal{H}$.

A scalar annealed learning rate $\mu_0/t$ combined with the momentum parameter $\beta_{opt}$ ought to provide an effective learning rate asymptotically equal to the optimal learning rate $\mathcal{H}^{-1}$. This rate 1) is achieved *without* ever performing a matrix inversion on $\mathcal{H}$ and 2) is independent of the choice of $\mu_0$, subject to the restriction in the previous paragraph.

We have dispensed with the need to invert the Hessian, and we next dispense with the need to store it. First notice that, unlike its inverse, stochastic estimates of $\mathcal{H}$ are readily available, so we use a stochastic estimate in (4). Secondly according to (2) we do not require the matrix $\beta_{opt}$, but rather $\beta_{opt}$ times the last weight update. For both linear and non-linear networks this dispenses with the $\mathcal{O}(n^2)$ storage requirements. This algorithm, which we refer to as adaptive momentum, does not require explicit knowledge or inversion of the Hessian, and can be implemented very efficiently as is shown in the next section.

## 2    Implementation

The algorithm we propose is

$$\omega_{t+1} = \omega_t + \mu_t G(\omega_t, x_t) + (I - \mu_0 \hat{\mathcal{H}}_t) \Delta\omega_t \tag{5}$$

where $\Delta\omega_t \equiv \omega_t - \omega_{t-1}$ and $\hat{\mathcal{H}}_t$ is a stochastic estimate of the Hessian at time $t$.

We first consider a single layer feedforward linear network. Since the weights connecting the inputs to different outputs are independent of each other we need only discuss the case for one output node. Each output node is then treated identically.

For one output node and $N$ inputs, the Hessian is $\mathcal{H} = \langle xx^T \rangle_x \in \mathbb{R}^{N \times N}$ where $\langle \cdot \rangle_x$ indicates expectation over the inputs $x$ and where $x^T$ is the transpose of $x$. The *single*-step estimate of the hessian is then just $\hat{\mathcal{H}}_t = x_t x_t^T$. The momentum term becomes

$$(I - \mu_0 \hat{\mathcal{H}}_t) \Delta\omega_t = (I - \mu_0(x_t x_t^T))\Delta\omega_t = \Delta\omega_t - \mu_0 x_t(x_t^T \Delta\omega_t). \tag{6}$$

Written in this way, we note that there is no matrix multiplication, just the vector dot product $x_t^T \Delta\omega_t$ and vector addition that are both $\mathcal{O}(n)$. For $M$ output nodes, the algorithm is then $\mathcal{O}(N_\omega)$ where $N_\omega = NM$ is the total number weights in the network.

For nonlinear networks the problem is somewhat more complicated. To compute $\hat{\mathcal{H}}_t \Delta\omega_t$ we use the algorithm developed by Pearlmutter [12] for computing the product of the hessian times an arbitrary vector.[4] The equivalent of one forward-back

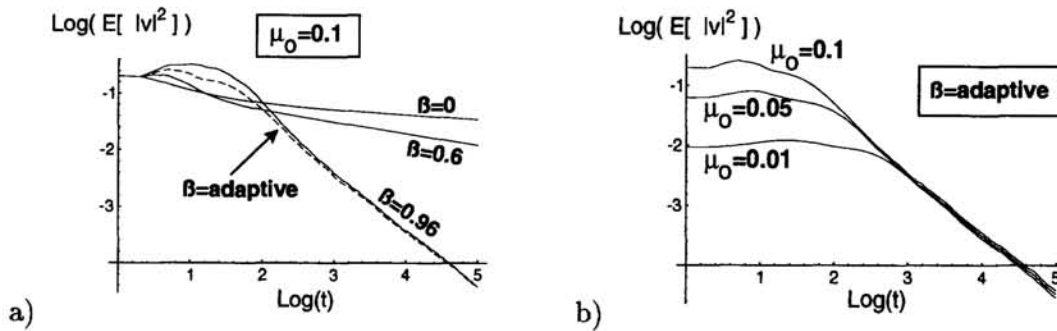

Figure 1: 2-D LMS Simulations: Behavior of $\log(E[|v|^2])$ over an ensemble of 1000 networks with $\lambda_1 = .4$ and $\lambda_1 = 4$, $\sigma_\epsilon^2 = 1$. a) $\mu_0 = 0.1$ with various $\beta$. Dashed curve corresponds to adaptive momentum. b) $\beta$ adaptive for various $\mu_0$.

propagation is required for this calculation. Thus, to compute the entire weight update requires two forward-backward propagations, one for the gradient calculation and one for computing $\hat{\mathcal{H}}_t \Delta \omega_t$.

The only constraint on $\mu_0$ is that $\mu_0 < 1/\lambda_{max}$. We use the on-line algorithm developed by LeCun, Simard, and Pearlmutter [13] to find the largest eigenvalue prior to the start of training.

## 3 Examples

In the following two subsections we examine the behavior of annealed learning with adaptive momentum on networks previously trained to a point close to an optimum, where the noise dominates. We look at very simple linear nets, large linear nets, and a large nonlinear net. In section 3.3 we couple adaptive momentum with automatic switching from constant to annealed learning.

### 3.1 Linear Networks

We begin with a simple 2-D LMS network. Inputs $x_t$ are gaussian distributed with zero mean and the targets $d$ at each timestep $t$ are $d_t = \omega_*^T x_t + \epsilon_t$ where $\epsilon_t$ is zero mean gaussian noise, and $\omega_*$ is the optimal weight vector. The weight error at time $t$ is just $v \equiv \omega_t - \omega_*$.

Figure 1 displays results for both constant and adaptive momentum with averages computed over an ensemble of 1000 networks. Figure (1a) shows the decay of $E[|v|^2]$ for $\mu_0 = 0.1$ and various values of $\beta$. As momentum is increased, the convergence rate increases. The optimal *scalar* momentum parameter is $\beta \equiv (1 - \mu_0 \lambda_{min}) = .96$. Adaptive momentum achieves essentially the same rate of convergence *without* prior knowledge of the Hessian.

Figure 1b shows the behavior of $E[|v|^2]$ for various $\mu_0$ when adaptive momentum is used. One can see that after a few hundred iterations the value of $E[|v|^2]$ is independent of $\mu_0$ (in all cases $\mu_0 < 1/\lambda_{max} < \mu_{\text{crit}}$).

Figure 2 shows the behavior of the misadjustment (mean squared error in excess of the optimum) for a 4-D LMS problem with a large condition number $\rho \equiv \lambda_{max}/\lambda_{min} = 10^5$. We compare 3 cases: 1) the *optimal learning rate matrix* $\mu_0 = \mathcal{H}^{-1}$ without momentum, 2) $\mu_0 = .5$ with the *optimal constant momentum matrix* $\beta = I - \mu_0 \mathcal{H}$, and 3) $\mu_0 = .5$ with the *adaptive momentum*. All three cases show similar behavior, showing the efficacy with which the *matrix* momentum

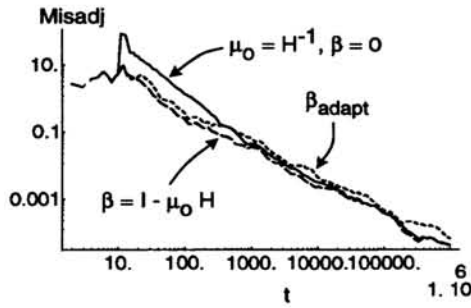
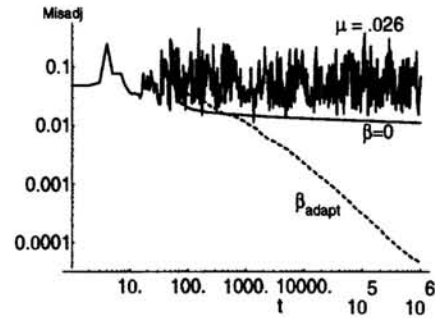

Figure 2: 4-D LMS with $\rho = 10^5$: Plot displays misadjustment. Annealing starts at $t = 10$. For $\beta_{adapt}$ and $\beta = I - \mu_0 \mathcal{H}$, we use $\mu_0 = .5$. Each curve is an average of 10 runs.

Figure 3: Linear Prediction: $\mu_0 = 0.26$. Curves show constant learning rate, annealing started at $t = 50$ without momentum, and with adaptive momentum.

mocks up the optimal learning rate matrix $\mu_0 = \mathcal{H}^{-1}$, and lending credence to the stochastic estimate of the Hessian used in adaptive momentum.

We next consider a large linear prediction problem (128 inputs, 16 outputs and eigenvalues ranging from $1.06 \times 10^{-5}$ to $19.98$ – condition number $\rho = 1.9 \times 10^6$)[5]. Figure 3 displays the misadjustment for 1) annealed learning with $\beta = \beta_{adapt}$, 2) annealed learning with $\beta = 0$, and 3) constant learning rate (for comparison purposes). As before, we have first trained (not shown completely) at constant learning rate $\mu_0 = .026$ until the MSE and the weight error have leveled out. As can be seen $\beta_{adapt}$ does much better than annealing without momentum.

### 3.2 Phoneme Classification

We next use phoneme classification as an example of a large nonlinear problem. The database consists of 9000 phoneme vectors taken from 48 50-second speech monologues. Each input vector consists of 70 PLP coefficients. There are 39 target classes. The architecture was a standard fully connected feedforward network with 71 (includes bias) input nodes, 70 hidden nodes, and 39 output nodes for a total of 7700 weights.

We first trained the network with constant learning rate until the MSE flattened out. At that point we either annealed without momentum, annealed with adaptive momentum, or used ASTC (which attempts to adjust $\mu_0$ to be above $\mu_{crit}$ – see next section). When annealing was used without momentum, we found that the noise went away, but the percent of correctly classified phonemes did not improve. Both the adaptive momentum and ASTC resulted in significant increases in the percent correct, however, adaptive momentum was significantly better than ASTC. In the next section, we examine this problem in more detail.

### 3.3 Switching on Annealing

A complete algorithm must choose an appropriate point to change from constant $\mu$ search to annealed learning. We use Moody and Darken's ASTC algorithm [4, 14] to accomplish this. ASTC measures the roughness of trajectories, switching to $1/t$ annealing when the trajectories become very rough – an indication that the noise in the updates is dominating the algorithm's behavior. In an attempt to satisfy

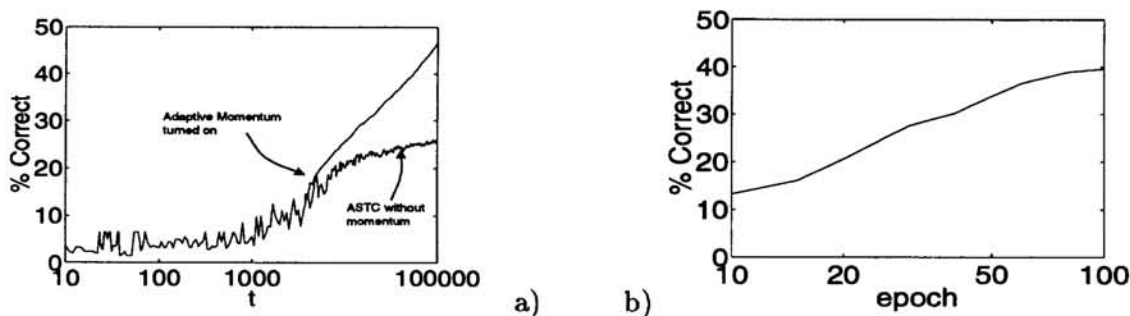

Figure 4: Phoneme Classification: Percent Correct a) ASTC without momentum (bottom curve) and adaptive momentum (top) as function of the number of input presentations. b) Conjugate Gradient Descent – one epoch equals one pass through the data, i.e. 9000 input presentations.

$\mu_0 > \mu_{crit}$, ASTC can also switch back to constant learning when trajectories become too smooth.

We return to the phoneme problem using three different training methods: 1) ASTC without momentum (with switching back and forth between annealed and constant learning), 2) adaptive momentum with annealing turned on when ASTC first suggests the transition (but no subsequent return to constant learning rate), and 3) standard conjugate gradient descent.

Figure 4a compares ASTC (no momentum) with adaptive momentum (using ASTC to turn on annealing). After annealing is turned on, the classification accuracy improves far more quickly with adaptive momentum.

Figure 4b displays the classification performance as a function of epoch using conjugate gradient descent (CGD). After 100 passes through the 9000 example dataset (900,000 presentations), the classification accuracy is 39.6%, or 7% below adaptive momentum's performance at 100,000 presentations. Note also that adaptive momentum is continuing to improve the optimization, while the ASTC and conjugate gradient descent curves have flattened out.

The cpu time used for the optimization was about the same for the CGD and adaptive momentum algorithms. It thus appears that our implementation of adaptive momentum costs about 9 times as much per pattern as CGD. We believe that the performance can be improved. Our complexity analysis [8] predicts a 3:1 cost ratio, rather than 9:1, and optimization comparable to that applied to the CGD code[6] should enhance the run-time performance of CGD.

For this problem, the performance of the two algorityms on the test set (no shown on graph) is not much different (31.7% for CGD versus 33.4% for adaptive momentum. Howver we are concerned here with the efficiency of the optimization, not generalization performance. The latter depends on dataset size and regularization techniques, which can easily be combined with any optimizer.

## 4 Summary

We have presented an efficient $\mathcal{O}(n)$ stochastic algorithm with few adjustable parameters that achieves fast convergence during the converge phase for both linear and nonlinear problems. It does this by incorporating curvature information without

explicit computation of the Hessian. We also combined it with a method (ASTC) for detecting when to make the transition between search and converge regimes.

## Acknowledgments

The authors thank Yann LeCun for his helpful critique. This work was supported by EPRI under grant RP8015-2 and AFOSR under grant FF4962-93-1-0253.

## Footnotes

[1]We assume that the inputs are i.i.d. This is achieved by random sampling with replacement from the training data.

[2]Venter [6] proposed a 1-D algorithm for optimizing the convergence rate that estimates the Hessian by time averaging finite differences of the gradient and scaling the learning rate by the inverse. Its extension to multiple dimensions would require $\mathcal{O}(n^2)$ storage and $\mathcal{O}(n^3)$ time for inversion. Both are prohibitive for large models.

[3]We refer to (4) as a heuristic since we have no theoretical results on the dynamics of the squared weight error for learning with this matrix of momentum parameters.

[4]We actually use a slight modification that calculates the *linearized* Hessian times a vector: $Df \otimes Df \, \Delta\omega_t$ where $Df$ is the Jacobian of the network output (vector) with respect to the weights, and $\otimes$ indicates a tensor product.

[5]Prediction of a $4 \times 4$ block of image pixels from the surrounding 8 blocks.

[6]CGD was performed using *nopt* written by Etienne Barnard and made available through the Center for Spoken Language Understanding at the Oregon Graduate Institute.

## References

[1] Genevieve B. Orr and Todd K. Leen. Weight space probability densities in stochastic learning: II. Transients and basin hopping times. In Giles, Hanson, and Cowan, editors, *Advances in Neural Information Processing Systems, vol. 5*, San Mateo, CA, 1993. Morgan Kaufmann.

[2] William Finnoff. Diffusion approximations for the constant learning rate backpropagation algorithm and resistence to local minima. In Giles, Hanson, and Cowan, editors, *Advances in Neural Information Processing Systems, vol. 5*, San Mateo, CA, 1993. Morgan Kaufmann.

[3] Larry Goldstein. Mean square optimality in the continuous time Robbins Monro procedure. Technical Report DRB-306, Dept. of Mathematics, University of Southern California, LA, 1987.

[4] Christian Darken and John Moody. Towards faster stochastic gradient search. In J.E. Moody, S.J. Hanson, and R.P. Lipmann, editors, *Advances in Neural Information Processing Systems 4*. Morgan Kaufmann Publishers, San Mateo, CA, 1992.

[5] Halbert White. Learning in artificial neural networks: A statistical perspective. *Neural Computation*, 1:425–464, 1989.

[6] J. H. Venter. An extension of the robbins-monro procedure. *Annals of Mathematical Statistics*, 38:117–127, 1967.

[7] Todd K. Leen and Genevieve B. Orr. Optimal stochastic search and adaptive momentum. In J.D. Cowan, G. Tesauro, and J. Alspector, editors, *Advances in Neural Information Processing Systems 6*, San Francisco, CA., 1994. Morgan Kaufmann Publishers.

[8] Genevieve B. Orr. *Dynamics and Algorithms for Stochastic Search*. PhD thesis, Oregon Graduate Institute, 1996.

[9] Mehmet Ali Tugay and Yalcin Tanik. Properties of the momentum LMS algorithm. *Signal Processing*, 18:117–127, 1989.

[10] John J. Shynk and Sumit Roy. Analysis of the momentum LMS algorithm. *IEEE Transactions on Acoustics, Speech, and Signal Processing*, 38(12):2088–2098, 1990.

[11] W. Wiegerinck, A. Komoda, and T. Heskes. Stochastic dynamics of learning with momentum in neural networks. *Journal of Physics A*, 27:4425–4437, 1994.

[12] Barak A. Pearlmutter. Fast exact multiplication by the hessian. *Neural Computation*, 6:147–160, 1994.

[13] Yann LeCun, Patrice Y. Simard, and Barak Pearlmutter. Automatic learning rate maximization by on-line estimation of the hessian's eigenvectors. In Giles, Hanson, and Cowan, editors, *Advances in Neural Information Processing Systems, vol. 5*, San Mateo, CA, 1993. Morgan Kaufmann.

[14] Christian Darken. *Learning Rate Schedules for Stochastic Gradient Algorithms*. PhD thesis, Yale University, 1993.